# Illumination-Invariant Face Recognition with a Contrast Sensitive Silicon Retina

**Joachim M. Buhmann**
Rheinische Friedrich–Wilhelms–Universität
Institut für Informatik II, Römerstraße 164
D-53117 Bonn, Germany

**Martin Lades**
Ruhr-Universität Bochum
Institut für Neuroinformatik
D-44780 Bochum, Germany

**Frank Eeckman**
Lawrence Livermore National Laboratory
ISCR, P.O.Box 808, L-426
Livermore, CA 94551

## Abstract

Changes in lighting conditions strongly effect the performance and reliability of computer vision systems. We report face recognition results under drastically changing lighting conditions for a computer vision system which concurrently uses a contrast sensitive silicon retina and a conventional, gain controlled CCD camera. For both input devices the face recognition system employs an elastic matching algorithm with wavelet based features to classify unknown faces. To assess the effect of analog on-chip preprocessing by the silicon retina the CCD images have been "digitally preprocessed" with a bandpass filter to adjust the power spectrum. The silicon retina with its ability to adjust sensitivity increases the recognition rate up to 50 percent. These comparative experiments demonstrate that preprocessing with an analog VLSI silicon retina generates image data enriched with *object-constant* features.

## 1 Introduction

Neural computation as an information processing paradigm promises to enhance artificial pattern recognition systems with the learning capabilities of the cerebral cortex and with the

adaptivity of biological sensors. Rebuilding sensory organs in silicon seems to be particularly promising since their neurophysiology and neuroanatomy, including the connections to cortex, are known in great detail. This knowledge might serve as a blueprint for the design of artificial sensors which mimic biological perception. Analog VLSI retinas and cochleas, as designed by Carver Mead (Mead, 1989; Mahowald, Mead, 1991) and his collaborators in a seminal research program, will ultimately be integrated in vision and communication systems for autonomous robots and other intelligent information processing systems.

The study reported here explores the influence of analog retinal preprocessing on the recognition performance of a face recognition system. Face recognition is a challenging classification task where object inherent distortions, like facial expressions and perspective changes, have to be separated from other image variations like changing lighting conditions. Preprocessing with a silicon retina is expected to yield an increased recognition rate since the first layers of the retina adjust their local contrast sensitivity and thereby achieve invariance to variations in lighting conditions.

Our face recognizer is equipped with a silicon retina as an adaptive camera. For comparison purposes all images are registered simultaneously by a conventional CCD camera with automatic gain control. Galleries with images of 109 different test persons each are taken under three different lighting conditions and two different viewing directions (see Fig. 1). These different galleries provide separate statistics to measure the sensitivity of the system to variations in light levels or contrast and image changes due to perspective distortions.

Naturally, the performance of an object recognition system depends critically on the classification strategy pursued to identify unknown objects in an image with the models stored in a database. The matching algorithm selected to measure the performance enhancing effect of retinal preprocessing deforms prototype faces in an elastic fashion (Buhmann et al., 1989; Buhmann et al., 1990; Lades et al., 1993). Elastic matching has been shown to perform well on the face classification task recognizing up to 80 different faces reliably (Lades et al., 1993) and in a translation, size and rotation invariant fashion (Buhmann et al., 1990). The face recognition algorithm was initially suggested as a simplified version of the *Dynamic Link Architecture* (von der Malsburg, 1981), an innovative neural classification strategy with fast changes in the neural connectivity during recognition stage. Our recognition results and conclusions are expected to be qualitatively typical for a whole range of face/object recognition systems (Turk, Pentland, 1991; Yuille, 1991; Brunelli, Poggio, 1993), since any image preprocessing with emphasis on object constant features facilitates the search for the correct prototype.

## 2   The Silicon Retina

The silicon retina used in the recognition experiments models the interactions between receptors and horizontal cells taking place in the outer plexiform layer of the vertebrate retina. All cells and their interconnections are explicitly represented in the chip so that the following description simultaneously refers to both biological wetware and silicon hardware. Receptors and horizontal cells are electrically coupled to their neighbors. The weak electrical coupling between the receptors smoothes the image and reduces the influence of voltage offsets between adjacent receptors. The horizontal cells have a strong lateral electrical coupling and compute a local background average. There are reciprocal excitatory-inhibitory synapses between the receptors and the horizontal cells. The horizontal cells use shunting inhibition to adjust the membrane conductance of the receptors and

thereby adjust their sensitivity locally. This feedback interaction produces an antagonistic center/surround organization of receptive fields at the output. The center is represented by the weakly coupled excitatory receptors and the surround by the more strongly coupled inhibitory horizontal cells. The center/surround organization removes the average intensity and expands the dynamic range without response compression. Furthermore, it enhances edges.

In contrast to this architecture, a conventional CCD camera can be viewed as a very primitive retina with only one layer of non-interacting detectors. There is no DC background removal, causing potential over- and underexposure in parts of the image which reduces the useful dynamic range. A mechanical iris has to be provided to adjust the mean luminance level to the appropriate setting. Since cameras are designed for faithful image registration rather than vision, on-chip pixel processing, if provided at all, is used to improve the camera resolution and signal-to-noise ratio.

Three adjustable parameters allow us to fine tune the retina chip for an object recognition experiment: (i) the diffusivity of the cones (ii) the diffusivity of the horizontal cells (iii) the leak in the horizontal cell membrane. Changes in the diffusivities affect the shape of the receptive fields, e.g., a large diffusivity between cones smoothes out edges and produces a blurred image. The other extreme of large diffusivity between horizontal cells pronounces edges and enhances the contrast gain. The retina chip has a resolution of 90 × 92 pixels, it was designed by (Boahen, Andreou, 1992) and fabricated in $2\mu$m n-well technology by MOSIS.

## 3   Elastic Matching Algorithm for Face Recognition

Elastic matching is a pattern classification strategy which explicitly accounts for local distortions. A prototype template is elastically deformed to measure local deviations from a new, unknown pattern. The amount of deformation and the similarity of local image features provide us with a decision criterion for pattern classification. The rubbersheet-like behavior of the prototype transformation makes elastic matching a particularly attractive method for face recognition where ubiquitous local distortions are caused for example by perspective changes and different facial expressions. Originally, the technique was developed for handwritten character recognition (Burr, 1981). The version of elastic matching employed for our face recognition experiments is based on attributed graph matching. A detailed description with a plausible interpretation in neural networks terms is published in (Lades *et al.*, 1993). Each prototype face is encoded as a planar graph with feature vectors attached to the vertices of the graph and metric information attached to the edges. The feature vectors extract local image information at pixel $\vec{x}_i$ in a multiscale fashion, i.e., they are functions of wavelet coefficients. Each feature vector establishes a correspondence between a vertex $i$ of a prototype graph and a pixel $\vec{x}_i$ in the image. The components of a feature vector are defined as the magnitudes of the convolution of an image with a set of two-dimensional, DC free Gaussian kernels centered at pixel $\vec{x}_i$. The kernels with the form

$$\psi_{\vec{k}}\left(\vec{x}\right) = \frac{\vec{k}^2}{\sigma^2} \exp\left(-\frac{\vec{k}^2 \vec{x}^2}{2\sigma^2}\right) \left[\exp\left(i\vec{k}\vec{x}\right) - \exp\left(-\sigma^2/2\right)\right] \tag{1}$$

are parameterized by the wave vector $\vec{k}$ defining their orientations and their sizes. To construct a self-similar set of filter functions we select eight different orientations and five

different scales according to

$$\vec{k}(\nu,\mu) = \frac{\pi}{2} \, 2^{-\nu/2} \left( \cos(\frac{\pi}{8}\mu), \sin(\frac{\pi}{8}\mu) \right) \tag{2}$$

with $\nu \in \{0,\ldots,4\}; \mu \in \{0,\ldots,7\}$. The multi-resolution data format represents local distortions in a robust way, i.e., only feature vectors in the vicinity $\vec{x}$ of an image distortion are altered by the changes. The edge labels encode metric information, in particular we choose the difference vectors $\Delta\vec{x}_{ij} \equiv \vec{x}_i - \vec{x}_j$ as edge labels.

To generate a new prototype graph for the database, the center of a new face is determined by matching a generic face template to it. A $7 \times 10$ rectangular grid with 10 pixel spacing between vertices and edges between adjacent vertices is then centered at that point. The saliency of image points is taken into account by deforming that generic grid so that each vertex is moved to the nearest pixel with a local maximum in feature vector length.

The classification of an unknown face as one of the models in the database or its rejection as an unclassified object is achieved by computing matching costs and distortion costs. The matching costs are designed to maximize the similarity between feature vector $\vec{J}_i^M$ of vertex $i$ in the model graph $(M)$ and feature vector $J^I(\vec{x}_i)$ associated with pixel $\vec{x}_i$ in the new image $(I)$. The cosine of the angle between both feature vectors

$$S(\vec{J}^I(\vec{x}_i), \vec{J}_i^M) = \frac{\vec{J}^I(\vec{x}_i) \cdot \vec{J}_i^M}{\|\vec{J}^I(\vec{x}_i)\| \, \|\vec{J}_i^M\|}. \tag{3}$$

is suited as a similarity function for elastic matching since global contrast changes in images only scale feature vectors but do not rotate them. Besides maximizing the similarity between feature vectors the elastic matching algorithm penalizes large distortions. The distortion cost term is weighted by a factor $\lambda$ which can be interpreted as a prior for expected distortions. The combined matching cost function which is used in the face recognition system compromises between feature similarity and distortion, i.e, it minimizes the cost function

$$\mathcal{H}^M\left(\{\vec{x}_i^I\}\right) = \frac{\lambda}{2} \sum_{(i,j)} \left(\Delta\vec{x}_{ij}^I - \Delta\vec{x}_{ij}^M\right)^2 - \sum_i S\left(\vec{J}(\vec{x}_i^I), \vec{J}_i^M\right) \tag{4}$$

for the model $M$ in the face database with respect to the correspondence points $\{\vec{x}_i^I\}$. $\langle i,j\rangle$ in Eq. (4) denotes that index $j$ runs over the neighborhood of vertex $i$ and index $i$ runs over all vertices. By minimizing Eq. (4) the algorithm assigns pixel $\vec{x}_i^I$ in the new image $I$ to vertex $i$ in the prototype graph $M$. Numerous classification experiments revealed that a steepest descent algorithm is sufficient to minimize cost function (4) although it is non-convex and local minima may cause non-optimal correspondences with reduced recognition rates.

During a recognition experiment all prototype graphs in the database are matched to the new image. A new face is classified as prototype $A$ if $H^A$ is minimal and if the significance criterion

$$\mathcal{R} \equiv \frac{\langle\mathcal{H}\rangle - \mathcal{H}^A}{\Sigma_H} > \Theta . \tag{5}$$

is fulfilled. The average costs $\langle\mathcal{H}\rangle$ and their standard deviation $\Sigma_H$ are calculated excluding match $A$. This heuristic is based on the assumption that a new face image strongly

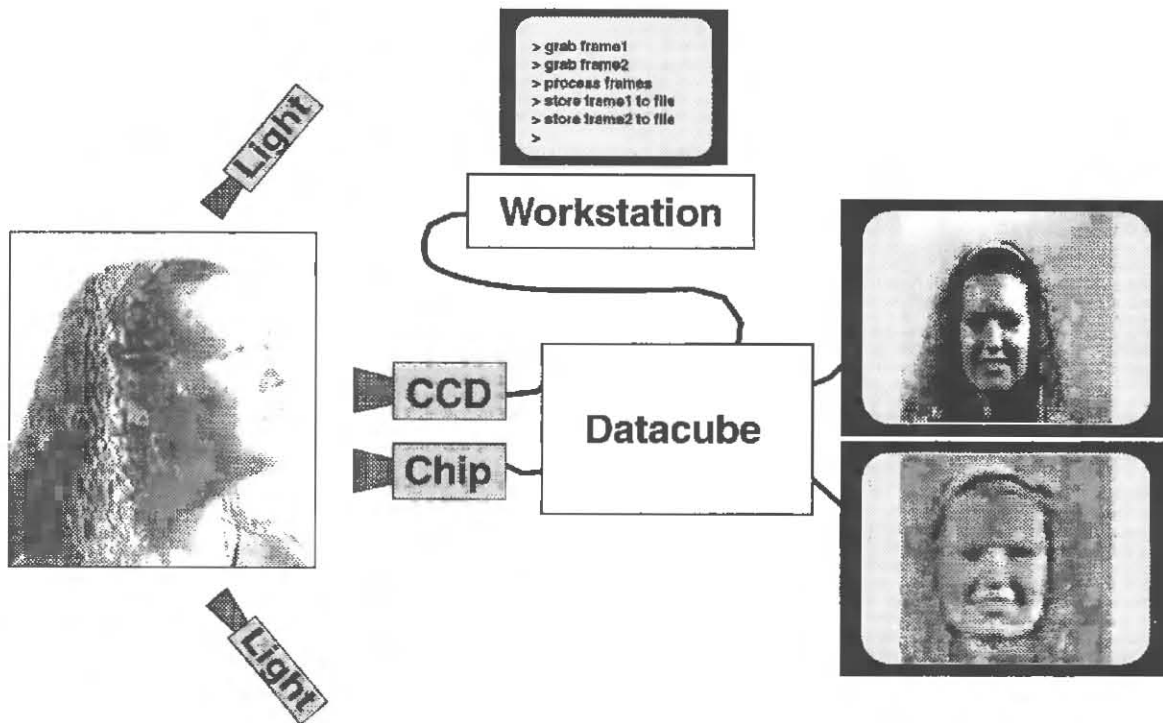

Figure 1: Laboratory setup of the face recognition experiments.

correlates with the correct prototype but the matching costs to all the other prototype faces is approximately Gaussian distributed with mean $\langle \mathcal{H} \rangle$ and standard deviation $\Sigma_H$. The threshold parameter $\Theta$ is used to limit the rate of false positive matches, i.e., to exclude significant matches to wrong prototypes.

## 4   Face Recognition Results

To measure the recognition rate of the face recognition system using a silicon retina or a CCD camera as input devices, pictures of 109 different persons are taken under 3 different lighting conditions and 2 different viewing directions. This setup allows us to quantify the influence of changes in lighting conditions on the recognition performance separate from the influence of perspective distortions. Figure 2 shows face images of one person taken under two different lighting setups. The images in Figs. 2a,c with both lights on are used as the prototype images for the respective input devices. To test the influence of changing lighting conditions the left light is switched off. The faces are now strongly illuminated from the right side. The CCD camera images (Figs. 2a,b) document the drastic changes of the light settings. The corresponding responses of the silicon retina shown in Figs. 2c,d clearly demonstrate that the local adaptivity of the silicon retina enables the recognition system to extract object structure from the bright and the dark side of the face. For control purposes all recognition experiments have been repeated with filtered CCD camera images. The filter was adjusted such that the power spectra of the retina chip images and the filtered CCD images are identical. The images (e,f) are filtered versions of the images (a,b). It is evident that information in the dark part of image (b) has been erased due to saturation effects of the CCD camera and cannot be recovered by any local filtering procedure.

We first measure the performance of the silicon retina under uniform lighting conditions,

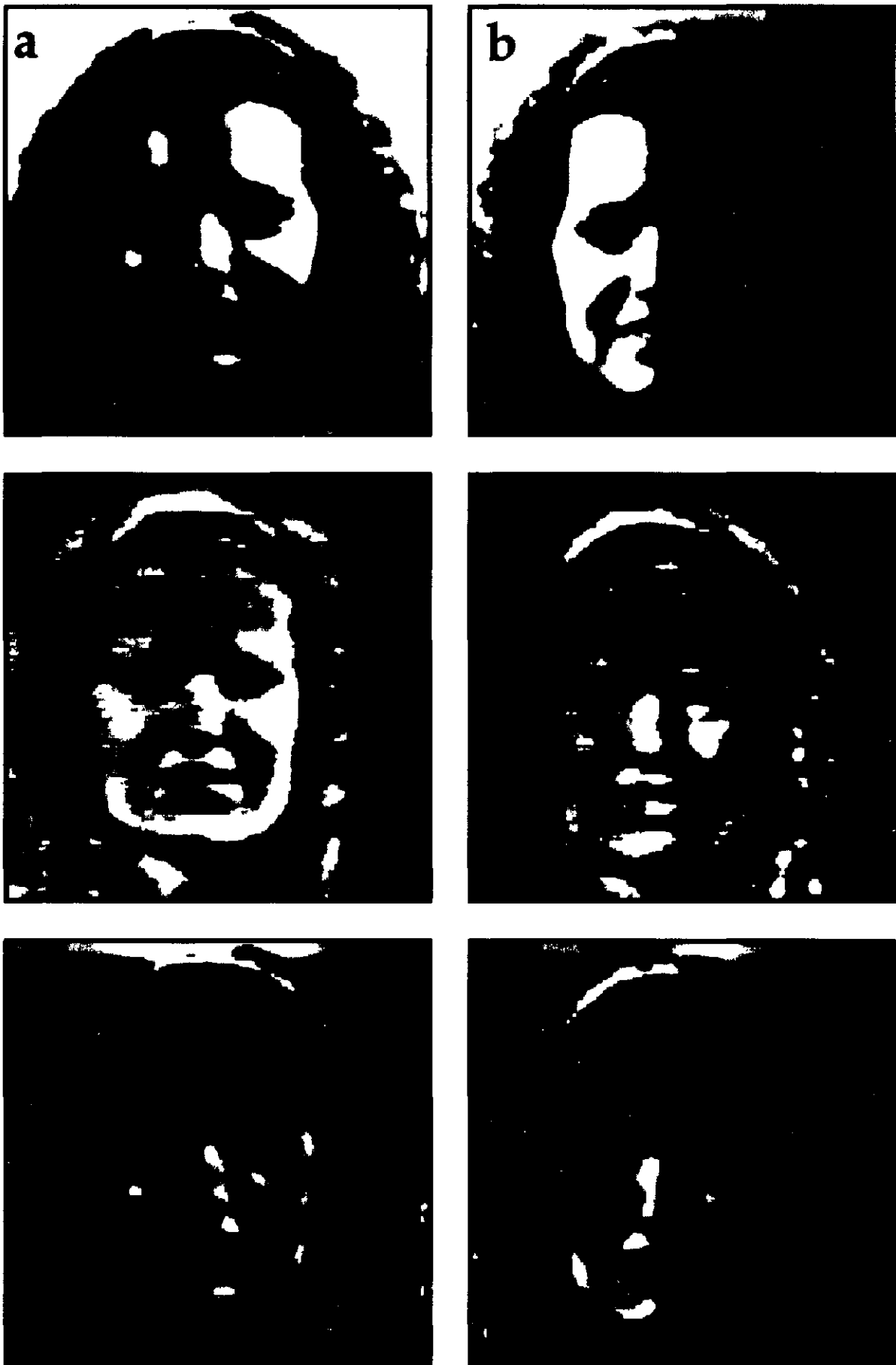

Figure 2: (a) Conventional CCD camera images (a,b) and silicon retina image (c,d) under different lighting conditions. The images (e,f) are filtered CCD camera images with a power spectrum adjusted to the images in (c,d). The images (a,c) are used to generate the

Table 1:  (a) Face recognition results in a well illuminated environment and (b) in an environment with drastic changes in lighting conditions.

|   | f. p. rate | silicon retina | conv. CCD | filt. CCD |
|---|---|---|---|---|
| **a** | 100% | 83.5 | 86.2 | 85.3 |
|   | 10% | 81.7 | 83.5 | 84.4 |
|   | 5% | 76.2 | 82.6 | 80.7 |
|   | 1% | 71.6 | 79.8 | 75.2 |
| **b** | 100% | 96.3 | 80.7 | 78.0 |
|   | 10% | 96.3 | 76.2 | 75.2 |
|   | 5% | 96.3 | 72.5 | 72.5 |
|   | 1% | 93.6 | 64.2 | 62.4 |

i.e., both lamps are on and the person looks 20–30 degrees to the right. The recognition system has to deal with perspective distortions only. A gallery of 109 faces is matched to a face database of the same 109 persons. Table 1a shows that the recognition rate reaches values between 80 and 90 percent if we accept the best match without checking its significance. Such a decision criterion is unpractical for many applications since it corresponds to a false positive rate (f.p. rate) of 100 percent. If we increase the threshold $\Theta$ to limit false positive matches to less than 1 percent the face recognizer is able to identify three out of four unknown faces. Filtering the CCD imagery does not hurt the recognition performance as the third column in Table 1a demonstrates. All necessary information for recognition is preserved in the filtered CCD images.

The situation changes dramatically when we switch off the lamp on the left side of the test person. We compare a test gallery of persons looking straight ahead, but illuminated only from the right side, to our model gallery. Table 1b summarizes the recognition results for different false positive rates. The advantage of using a silicon retina are 20 to 45 percent higher recognition rates than for a system with a CCD camera. For a false positive rate below one percent a silicon retina based recognition system identifies two third more persons than a conventional system. Filtering does not improve the recognition rate of a system that uses a CCD camera as can be seen in the third column.

Our comparative face recognition experiment clearly demonstrates that a face recognizer with a retina chip is performing substantially better than conventional CCD camera based systems in environments with uncontrolled, substantially changing lighting conditions. Retina-like preprocessing yields increased recognition rates and increased significance levels. We expect even larger discrepancies in recognition rates if object without a bilateral symmetry have to be classified. In this sense the face recognition task does not optimally explore the potential of adaptive preprocessing by a silicon retina. Imagine an object recognition task where the most significant features for discrimination are hardly visible or highly ambiguous due to poor illumination. High error rates and very low significance levels are an inevitable consequence of such lighting conditions.

The limited resolution and poor signal-to-noise ratio of silicon retina chips are expected to be improved by a new generation of chips fabricated in 0.7$\mu$m CMOS technology with a

potential resolution of 256 × 256 pixels. Lighting conditions as simulated in our recognition experiment are ubiquitous in natural environments. Autonomous robots and vehicles or surveillance systems are expected to benefit from the silicon retina technology by gaining robustness and reliability. Silicon retinas and more elaborate analog VLSI chips for low level vision are expected to be an important component of an *Adaptive Vision* System.

**Acknowledgement:** It is a pleasure to thank K. A. Boahen for providing us with the retina chips. We acknowledge stimulating discussions with C. von der Malsburg and C. Mead. This work was supported by the German Ministry of Science and Technology (ITR-8800-H1) and by the Lawrence Livermore National Laboratory (W-7405-Eng-48).

# References

Boahen, K., Andreou, A. 1992. A Contrast Sensitive Silicon Retina with Reciprocal Synapses. *Pages 764–772 of: NIPS91 Proceedings.* IEEE.

Brunelli, R., Poggio, T. (1993). Face Recognition: Features versus Templates. *IEEE Trans. on Pattern Analysis Machine Intelligence*, **15**, 1042–1052.

Buhmann, J., Lange, J., von der Malsburg, C. 1989. Distortion Invariant Object Recognition by Matching Hierarchically Labeled Graphs. *Pages I 155–159 of: Proc. IJCNN, Washington.* IEEE.

Buhmann, J., Lades, M., von der Malsburg, C. 1990. Size and Distortion Invariant Object Recognition by Hierarchical Graph Matching. *Pages II 411–416 of: Proc. IJCNN, SanDiego.* IEEE.

Burr, D. J. (1981). Elastic Matching of Line Drawings. *IEEE Trans. on Pat. An. Mach. Intel.*, **3**, 708–713.

Lades, M., Vorbrüggen, J.C., Buhmann, J., Lange, J., von der Malsburg, C., Würtz, R.P., Konen, W. (1993). Distortion Invariant Object Recognition in the Dynamic Link Architecture. *IEEE Transactions on Computers*, **42**, 300–311.

Mahowald, M., Mead, C. (1991). The Silicon Retina. *Scientific American*, **264**(5), 76.

Mead, C. (1989). *Analog VLSI and Neural Systems.* New York: Addison Wesley.

Turk, M., Pentland, A. (1991). Eigenfaces for Recognition. *J. Cog. Sci.*, **3**, 71–86.

von der Malsburg, Christoph. 1981. *The Correlation Theory of Brain Function.* Internal Report. Max-Planck-Institut, Biophys. Chem., Göttingen, Germany.

Yuille, A. (1991). Deformable Templates for Face Recognition. *J. Cog. Sci.*, **3**, 60–70.